# Probabilistic Conformal Distillation for Enhancing Missing Modality Robustness

**Mengxi Chen**[1,3]  **Fei Zhang**[1]  **Zihua Zhao**[1]  **Jiangchao Yao**[1,3†]  **Ya Zhang**[2,3], **Yanfeng Wang**[2,3]

[1] Cooperative Medianet Innovation Center, Shanghai Jiao Tong University
[2] School of Artificial Intelligence, Shanghai Jiao Tong University
[3] Shanghai Artificial Intelligence Laboratory
{mxchen_mc, ferenas, sjtuszzh, Sunarker, ya_zhang, wangyanfeng}@sjtu.edu.cn

## Abstract

Multimodal models trained on modality-complete data are plagued with severe performance degradation when encountering modality-missing data. Prevalent cross-modal knowledge distillation-based methods precisely align the representation of modality-missing data and that of its modality-complete counterpart to enhance robustness. However, due to the irreparable information asymmetry, this determinate alignment is too stringent, easily inducing modality-missing features to capture spurious factors erroneously. In this paper, a novel multimodal Probabilistic Conformal Distillation (PCD) method is proposed, which considers the inherent indeterminacy in this alignment. Given a modality-missing input, our goal is to learn the unknown Probability Density Function (PDF) of the mapped variables in the modality-complete space, rather than relying on the brute-force point alignment. Specifically, PCD models the modality-missing feature as a probabilistic distribution, enabling it to satisfy two characteristics of the PDF. One is the extremes of probabilities of modality-complete feature points on the PDF, and the other is the geometric consistency between the modeled distributions and the peak points of different PDFs. Extensive experiments on a range of benchmark datasets demonstrate the superiority of PCD over state-of-the-art methods. Code is available at: https://github.com/mxchen-mc/PCD.

## 1 Introduction

Classical multimodal learning [29, 20, 36, 3] typically pre-supposes that the modalities of all data are complete throughout both the training and testing. However, due to collection constraints such as device limitations, budget constraints, and restrained working conditions, it is challenging to guarantee such a perfect condition [47]. When modalities are partially available, the performance of models trained on modality-complete data will deteriorate remarkably. This thereby attracts a range of explorations contributed recently, given that multimodal learning is playing an increasing role.

The existing approaches to address this problem generally fall into two paradigms, i.e., independent modeling [11, 39, 7] and unified modeling [9, 13, 46] for different modality-missing combinations, of which the latter is preferred due to the merits of low-storage cost and flexibility. As one prevalent line of unified modeling, cross-modal knowledge distillation (KD) has achieved persistent advancements in recent years [40, 51, 47, 46]. It attempts to guide the modality-missing representation to align with its modality-complete counterpart, facilitating the training under the guidance of privileged modality-complete information. However, these methods fail to consider that once a modality is missing, it is impossible to recover its personalized information via a brute-force alignment, which

---

[†]The corresponding author is Jiangchao Yao (Sunarker@sjtu.edu.cn).

has been revealed theoretically by [18]. Roughly ignoring this inherent information asymmetry in the alignment can instead lead multimodal models to fit spurious factors erroneously.

We conjecture that when partial modalities are missing, the retaining information is merely correlated to that of modality-complete input in a probabilistic sense. Specifically, given a modality-missing input, the unknown Probability Density Function (PDF) of its mapped variables in the modality-complete space peaks at the corresponding modality-complete feature and diminishes when diverging away from this point, as illustrated in Figure 1 (b). Compared to previous deterministic methods, learning the PDF is a more reasonable and tolerant way to transfer privileged information. Although the closed form of the oracle PDF is unknown, we can approximate it by modeling a probabilistic distribution with two key

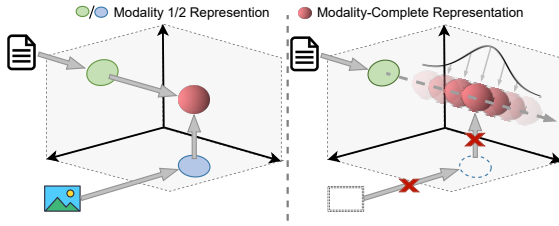

Figure 1: In a two-modality scenario, when both modalities are present, the modality-complete representation is derived through fusion. When one modality is absent, the mapped representation inferred from the remaining modality is subject to a certain probability distribution in the modality-complete space.

characteristics: (1) In a modeled distribution, the positive points closer to the modality-complete representation should demonstrate high probabilities and the negative points farther away should exhibit low probabilities. (2) For different distributions from distinct samples, the relation of their peak points should be conformal with that of their modality-complete representations. Here, the former focuses on extreme probability points, while the latter ensures geometric consistency.

With the above intuition, we propose a novel multimodal Probabilistic Conformal Distillation (PCD) method, which aims to align the modality-missing feature with its modality-complete counterpart probabilistically. Specifically, PCD parameterizes each modality-missing representation as an independent probabilistic distribution and optimizes it to satisfy the two characteristics. To achieve (1), the log probabilities of the distribution are maximized at positive points and minimized at negative points. To achieve (2), PCD introduces a contrastive-learning-based approach to align the geometric structure of the peak points of distributions with that of the modality-complete features. In this way, the modeled modality-missing distributions can approximate their corresponding PDFs, thereby facilitating the privileged modality-complete information transfer more efficiently.

In a nutshell, our contributions can be summarized as follows:

- We propose a multimodal Probabilistic Conformal Distillation method to handle the missing modality problem, which transfers privileged information of modality-complete representation by considering the indeterminacy in the mapping from incompleteness to completeness.

- We parameterize different modality-missing representations as distinct distributions to fit their unknown PDFs in the modality-complete space. This is specially realized by considering the probabilities of extreme points and ensuring the geometric consistency between peak points of different PDFs and modeled distributions.

- We conduct comprehensive experiments to demonstrate the effectiveness of PCD across a range of modality-missing scenarios. Extensive comparison on multimodal classification and segmentation tasks consistently validate the superior performance of our method compared to the state-of-the-art approaches. Particularly, PCD achieves an average improvement of about 5% for the seven modality-missing scenarios on the classification dataset CeFA.

## 2 Related Work

We roughly categorize recent explorations to improve the missing modality robustness into two paradigms: independent modeling methods and unified modeling methods.

### 2.1 Independent Modeling for Missing Modality

Many works address the modality-missing problem by training specific models for different modality-missing combinations [41, 10, 31, 26]. In a certain modality-missing case, some approaches recon-

struct the original data of the missing modalities from the available ones [2, 22, 28, 28]. However, the complexity of the data reconstruction usually leads to instability and may introduce noise to affect the main task [30, 52]. To alleviate this problem, many works try to reconstruct missing modalities at the representation level [11, 39, 7, 5]. Nevertheless, training specific models for each missing case tend to be inflexible and storage-consuming for real-world scenarios.

## 2.2 Unified Modeling for Missing Modality

Recently, there has been a growing interest in improving the robustness of unified multimodal models against a range of modality-missing combinations [56, 33, 21, 19]. To achieve this goal, some methods attempt to extract redundant information across modalities by designing different *fusion* networks [15, 53, 9, 50]. However, these methods ignore the complementary information, resulting in suboptimal performance to the specific models. Other methods capture the comprehensive information through dynamical fusion strategies [13, 14, 12, 6]. To be specific, these methods utilize uncertainty estimation techniques to learn the dynamical strength relationships among modalities within different samples, allowing for the adaptive assignment of weights to each available modality. To harness both redundant and complementary information of available modalities more effectively, some methods [32, 51, 47, 46] introduce a *distillation* loss to guide the unified model to imitate representations or inter-sample relations of the modality-complete model. This distillation process help the unified model acquire additional privileged information from complete modalities, so as to improve multimodal robustness [44, 43, 45, 42]. However, previous KD-based methods often emphasize precisely aligning the modality-missing representation with its complete counterpart, which probably causes the overfitting on spurious features due to the inherent information asymmetry.

# 3 Method

## 3.1 Preliminary

**Notations.** Suppose that we have a modality-complete training set of $\{(\mathrm{x}_i^\star, y_i)\}_{i=1}^N$, where each input $\mathrm{x}_i^\star$ comprises $M$ modalities, denoted as $\mathrm{x}_i^\star = \{x_i^m\}_{m=1}^M$, and $y_i$ represents the corresponding ground-truth label. $N$ is the dataset size. Our goal is to train a unified model capable of accurately predicting the label $y_i$ for any modality-missing case $\mathrm{x}_i \subseteq \mathrm{x}_i^\star$ & $\mathrm{x}_i \neq \varnothing$. Here, we use an auxiliary indicator vector $\delta_i$ for $\mathrm{x}_i$, where $\forall m, \delta_i^m \in \{0, 1\}$ indicates the modality in $\mathrm{x}_i$ missing or not. During testing, we construct different modality-missing cases to comprehensively evaluate the robustness.

**Motivation.** Owing to the inherent information asymmetry, modality-complete and modality-missing representations cannot be perfectly aligned, even with redundant information. This claim is experientially supported by the results in Appendix D. Therefore, we try to align the representation of modality-missing input $\mathrm{x}_i$ with that of modality-complete input $\mathrm{x}_i^\star$ in a probabilistic sense. As shown in the right panel of Figure 1, we conjecture that the representation $z_i$ of modality-missing input $\mathrm{x}_i$ has a probabilistic peak expectation towards the representation $z_i^\star$ of the modality-complete input $\mathrm{x}_i^\star$. In other words, the corresponding PDF $p(z_i|\mathrm{x}_i)$ satisfies the following requirement

$$z_i^\star = \arg\max_{z_i \in Z} p(z_i|\mathrm{x}_i), \tag{1}$$

where $Z$ denotes the representation space. Generally, approximating the unknown PDF $p(z_i|\mathrm{x}_i)$ is a more relaxed condition compared with the stringent point alignment in previous KD-based methods.

## 3.2 Probabilistic Conformal Distillation

### 3.2.1 Objective

Although $p(z_i|\mathrm{x}_i)$ is unknown, even about the function family of the distribution, we can define an easier distribution $q(z_i|\mathrm{x}_i)$ to approximate its characteristics. Specifically, we can force $q(z_i|\mathrm{x}_i)$ to follow the two-fold characteristics: 1) *extremum property*. In a modeled distribution $q(z_i|\mathrm{x}_i)$, positive points near the modality-complete representation $z_i^\star$ should exhibit higher probabilities, and negative points distant from $z_i^\star$ approach far smaller probabilities. (2) *conformal property*. Given different samples, the relationship of the peak points of $q(z|\mathrm{x})$ should be conformal with that of their corresponding modality-complete points $z^\star$.

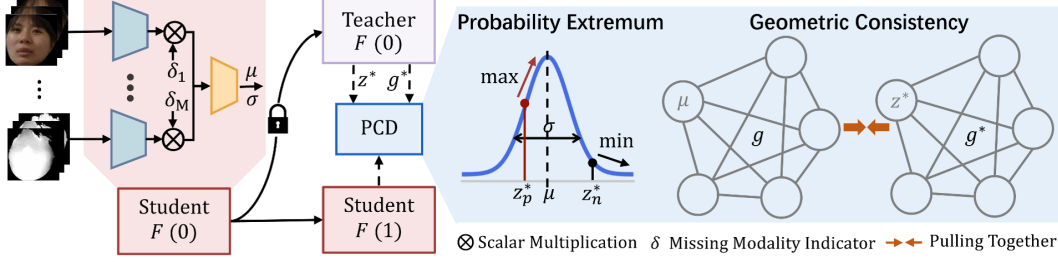

Figure 2: An overview of the proposed method. PCD is a self-KD architecture, where the teacher and student share the same framework. The teacher provides the modality-complete feature $z^\star$ and the geometric structure $g^\star$ to guide the student. In the student, modality-missing features are parameterized as different normal distributions to fit the corresponding PDF. To achieve this, PCD maximizes distributions at positive $z_p^\star$ and minimizes it at $z_n^\star$, while aligning $g$ with positive $g^\star$.

To achieve the former property, we first define a positive set $Z_p$ which includes all modality-complete representations $z_p^\star$ that are close to $z_i^\star$, and a negative set $Z_n$, consisting of the remaining representations $z_n^\star$ that are far away from $z_i^\star$. For example, in a classification task, $Z_p$ contains all $z_p^\star$ of the same class as $x_i$, while $Z_n$ consists of $z_n^\star$ from other classes. In a segmentation, $Z_p$ contains only $z_i^\star$. Then, the following characteristic should be satisfied

$$q(z_p^\star \in Z_p | x_i) \gg q(z_n^\star \in Z_n | x_i) \approx 0. \tag{2}$$

Equation (2) encourages that the probability of any one positive point $z_p^\star$ to be greater than the probabilities of all negative points $z_n^\star \in Z_n$, which helps to satisfy the extremum property.

Regarding the conformal property, let $g_i$ denote the geometric vector for $z_i$. Each element in $g_i$ calculates the distance between the peak points of $q(z_i | x_i)$ and other modeled distributions $q(z_j | x_j)$. Vector $g^\star$ represents the geometric distance calculated by the modality-complete representations $z^\star$ in the same manner. Similar to $Z_p$ and $Z_n$, we use $G_p$ and $G_n$ to include the positive and negative geometric vectors respectively. The set $G_p$ contains all the vectors $g_p^\star$ corresponding to $z_p^\star$ and the same relation applies to $G_n$ and $z_n^\star$. Then pursue the following characteristic satisfied

$$s(g_p^\star \in G_p, g_i) \gg s(g_n^\star \in G_n, g_i), \tag{3}$$

where $s(\cdot, \cdot) > 0$ is one of the metrics for measuring the vector similarity. Equation (3) hopes the similarity between the geometric vector $g_i$ and any positive vector $g^\star$ to be larger than that between $g_i$ and negative vectors $g_n^\star \in G_n$. To meet the characteristics in Equation (2) and Equation (3), we propose a probabilistic conformal objective to optimize $q(z | x_i)$:

$$\max \frac{\prod_{g_p^\star \in G_p} s(g_p^\star, g_i) \prod_{z_p^\star \in Z_p} q(z_p^\star | x_i)}{\prod_{z_n^\star \in Z_n} q(z_n^\star | x_i)}. \tag{4}$$

Specifically in Equation (4), to satisfy the extremum property, we propose to maximize the probabilities of $q(z_i | x_i)$ at positive points $z_p^\star \in Z_p$ and minimize them at negative points $z_n^\star \in Z_n$. To achieve Equation (3), we introduce a contrastive learning-based approach to maximize the similarities $s(g_p^\star, g_i)$. Notice that, here the minimization of $s(g_n^\star, g_i)$ is not emphasized, since it is *implicitly included* in the contrastive-learning-based similarity. By simplifying Equation (4) with the log function, we can transform the objective function into a more manageable form, expressed as

$$\max \underbrace{\left( \sum_{z_p^\star \in Z_p} \log q(z_p^\star | x_i) - \sum_{z_n^\star \in Z_n} \log q(z_p^\star | x_i) \right)}_{\text{Probability Extremum}} + \underbrace{\sum_{g_p^\star \in G_p} \log s(g_p^\star, g_i)}_{\text{Geometric Consistency}}. \tag{5}$$

Equation (5) consists of two parts, where the first term focuses on extreme probability points, while the second term is for the geometric consistency. In the following, we introduce the implementation of Equation (5) on how to model the modality-missing distributions (Section 3.2.2) and fit the corresponding PDFs (Section 3.2.3).

### 3.2.2 Multimodal Probabilistic Modeling.

The framework of PCD is shown in Figure 2. For each modality-missing input $x_i$, we establish an individual D-dimensional normal distribution $q(z_i|x_i)$, with its mean and variance directly determined as by the multimodal encoder follows

$$q(z_i|x_i) \sim \mathcal{N}\left(z_i; \mu_i, \sigma_i^2\right), \text{ where } \mu_i = f(x_i), \sigma_i = h(\mu_i). \tag{6}$$

The features $\mu_i \in \mathbb{R}^D, \sigma_i \in \mathbb{R}^D$ represent the mean and variance vectors of the multimodal distribution $\mathcal{N}\left(\mu_i, \sigma_i^2\right)$, respectively. $f(\cdot)$ denotes the multimodal encoder, while $h(\cdot)$ is the head module for computing the variance vectors. We maximize Equation (5) for each modality-missing distribution $q(z_i|x_i)$ to fit the corresponding PDF. The probabilistic modeling maps each modality-missing input $x_i$ to a density region in the representation space, rather than a single deterministic vector point, which enhances the tolerance to lower-quality modality-missing data and prevents the multimodal encoder from affecting representation capacity by learning some spurious factors.

### 3.2.3 Probabilistic Conformal Distillation

After modeling the modality-missing input as a Gaussian distribution $q(z_i|x_i)$, we aim to approximate $q(z_i|x_i)$ to the unknown PDF $p(z_i|x_i)$ to transfer the modality-complete information. This is accomplished by optimizing two terms in Equation (5), that is, the probability extremum term and the geometric consistency term.

**Probability Extremum.** The probability extremum term in Equation (5) enables $q(z_i|x_i)$ to have higher probabilities at positive points in $Z_p$ and lower probabilities at negative points in $Z_n$. By inserting the Gaussian function into the probability extremum term and eliminating the constant, the extremum term can be maximized by minimizing its negative form, namely, the following loss,

$$\mathcal{L}_u = \sum_{\{p|y_p=y_i\}} \sum_d \left(\frac{(z_{p,d}^\star - \mu_{i,d})^2}{2(\sigma_{i,d})^2} + \log \sigma_{i,d}\right) - \sum_{\{n|y_n \neq y_i\}} \sum_d \left(\frac{(z_{n,d}^\star - \mu_{i,d})^2}{2(\sigma_{i,d})^2} + \log \sigma_{i,d}\right). \tag{7}$$

Prior works [4, 35] in high-dimensional latent distribution learning report that the variance collapse is a commonly encountered issue. This phenomenon typically occurs because the network is encouraged to predict small $\sigma$ values to mitigate the unstable gradients that arise while using Stochastic Gradient Descent. To prevent this problem, we empirically implement a clipping operation on Equation (7), stopping the optimization when $\sigma$ becomes too small. For brevity, we focus on analyzing the first half of Equation (7). Its optimization is carried out in two aspects: (1) minimizing the distance between the mean $\mu_{i,d}$ and the positive modality-complete representations $z_{p,d}^\star$ of the teacher, *i.e.*, $(z_{p,d}^\star - \mu_{i,d})^2$; (2) correlating this distance with $\sigma_{i,d}^2$, where larger distances correspond to higher variance, and vice versa. This relationship allows us to estimate the element-wise quality of each mean vector $\mu_i$, where the closer proximity to $z_{p,d}^\star$ signifies more information contained.

**Geometric Consistency.** The geometric consistency term aims to align the structure vector $g_i$ with its positive counterparts in $G_p$. Specifically, we represent the geometric vector $g^\star$ of PDFs by calculating the distances of their peak points $z^\star$, and $g$ is obtained by the distances of mean vectors $\mu$, namely:

$$g_i^\star(b) = \alpha(z_i^\star, z_b^\star), \ g_i(b) = \alpha(\mu_i, \mu_b),$$

where $g_i, g_i^\star$ are $|B|$-dimensional vectors with $\mu_i, z_i^\star$ as the cores, respectively. $|B|$ is the batch size. Theoretically, $\alpha(\cdot, \cdot)$ can be any formula for calculating the distance between vectors. For classification tasks, $\alpha(\cdot, \cdot)$ is the Euclidean distance. For segmentation tasks, since the dimension of the modality-missing and modality-complete features could be very high, we choose the inner product to mitigate the curse of dimensionality. Notice that $g_i, g_i^\star$ are computed across all samples in the batch, without distinguishing between positive and negative samples.

Like $Z_p$, the set $G_p$ contains the positive geometric vectors $g_p^\star$, whose core $z_p^\star$ share the same class as $x_i$, namely $G_p = \{g_p^\star | y_p = y_i\}$. For the similarity function $s(g_p^\star, g_i)$ in the geometric consistency term, we employ the following contrastive learning-based form:

$$s(g_p^\star, g_i) = \frac{\exp(\beta(g_p^\star, g_i)/\tau)}{\exp(\beta(g_p^\star, g_i)/\tau) + \sum_{\{n|y_n \neq y_i\}} \exp(\beta(g_n^\star, g_i)/\tau)}, \tag{8}$$

where $\beta(g, g^\star)$ calculates the cosine similarity between $g$ and $g^\star$, $\tau$ is the temperature coefficient. It is worth noting that in the segmentation task, due to the high dimensionality of multimodal features, only one negative vector is selected to conserve computational resources. Then, PCD aligns $g_i$ with $g^\star \in G_p$ through minimizing:

$$\mathcal{L}_g = - \sum_{\{p|y_p=y_i\}} \log s(g_p^\star, g_i), \tag{9}$$

To reiterate, the difference between the contrastive learning-based loss $\mathcal{L}_g$ in classification and segmentation tasks is analogous to that between supervised contrastive learning [23] and contrastive learning [16, 55, 54, 48]. The former considers all $g^\star$ sharing the same class as $g_i$ as positive samples, whereas the latter only uses $g_i^\star$ from the same instance as the positive sample.

By optimizing Equation (7) and Equation (9), each modality-missing distribution can fit the corresponding $p(z_i|x_i^m)$ and capture privileged information from the teacher in a more tolerant way.

### 3.3 Training Process

The framework of PCD, depicted in Figure 2, adopts a teacher-student architecture. Self-KD [24] is introduced to build an end-to-end distillation system, where the parameters of the fixed teacher $F(0)$ are obtained from the warm-up stage. During the training stage, the teacher model handles the modality-complete data and provides supervision for the student $F(1)$.

**Overall Loss.** The overall loss function is formulated as:

$$\mathcal{L} = \mathcal{L}_t + \lambda(\mathcal{L}_u + \mathcal{L}_g), \tag{10}$$

where $\lambda$ is the hyperparameter used to balance different losses, and the experiments show that $\lambda$ is insensitive in a certain range. $\mathcal{L}_t$ represents the task learning loss, which is defined by the specific primary task. For example, when the primary task is classification, $\mathcal{L}_t$ corresponds to the cross-entropy loss. The training procedure is shown in Algorithm 1 in Appendix A.

### 3.4 Discussion

PCD proposes to fit the PDFs of variables in the representation space by utilizing different parameterized Gaussian distributions. Compared to existing KD-based methods, PCD offers a more tolerant and reasonable way to transfer the privileged information from the modality-complete teacher to the modality-missing student. Specifically, it optimizes the probabilities of modeled distributions at extremum points and constrains the alignment between the geometric structures of teacher representations and the mean vectors of modeled distributions. Besides, regarding the complexity, PCD only introduces some head modules in the encoder to estimate the variance, which is lightweight and efficient and can be easily applied to many existing multimodal fusion methods.

## 4 Experiments

### 4.1 Experimental Setup

**Datasets.** We implement experiments on four multimodal datasets, comprising two classification datasets CASIA-SURF and CeFA, and two segmentation datasets NYUv2 and Cityscapes.

**CASIA-SURF** [49] and **CeFA** [27] are two large face anti-spoofing datasets that include samples across three modalities: RGB, Depth and infrared (IR). For CASIA-SURF [49], we adhere to the intra-testing protocol established by the authors, ensuring consistency and reliability in our experimental results. This dataset comprises 29,000 samples for training, 1,000 for validation, and 57,000 for testing. Similarly, in CeFA [27], we employ a cross-ethnicity and cross-attack protocol as recommended by the authors, which divides the dataset into training, validation, and testing sets with 35,000, 18,000, and 54,000 samples respectively.

**NYUv2** [37] and **Cityscapes** [8] are both two-modality segmentation datasets, each comprising RGB and Depth modalities. NYUv2 [37] contains a total of 1,449 indoor RGB-D images, with 795 designated for training and 654 for testing. NYUv2 employs a common 40-class label setting, facilitating comparative analysis across various segmentation algorithms. Cityscapes [8] is an outdoor

Table 1: Performance under different multimodal conditions, where "R", "D", and "I" respectively represent the available RGB, Depth, and IR modality. "Average" is the average performance over all the possible conditions. ACER ↓ means that the lower the ACER value, the better the performance, while mIOU ↑ is the opposite. The best results are in bold and the second-best ones are marked with underline. "Δ" means the performance gap between PCD and the best results.

| Method | CASIA-SURF (ACER ↓) | | | | | | | |
|---|---|---|---|---|---|---|---|---|
| | {R} | {D} | {I} | {R,D} | {R,I} | {D,I} | {R,D,I} | Average |
| Traditional [49] | 23.03 | 17.10 | 49.53 | 10.40 | 41.02 | 11.26 | 1.40 | 22.11 |
| Separate Model [49] | 10.01 | 4.45 | 11.65 | 3.41 | 6.32 | 3.54 | 1.23 | 5.80 |
| Augmentation [1] | 11.75 | 5.87 | 16.62 | 4.61 | 6.68 | 4.95 | 2.21 | 7.52 |
| HeMIS [15] | 14.36 | 4.70 | 16.21 | 3.23 | 6.27 | 3.68 | 1.97 | 7.18 |
| MMFormer [50] | 11.15 | 4.67 | 13.99 | 1.93 | 4.77 | 3.10 | 1.94 | 5.93 |
| MMANET [46] | 8.57 | 2.27 | 10.04 | 1.61 | 3.01 | 1.18 | 0.87 | 3.94 |
| MD [12] | 10.84 | 6.65 | 19.43 | 12.64 | 7.84 | 3.99 | 0.96 | 7.30 |
| ETMC [14] | 7.91 | 4.73 | 7.54 | 1.39 | 4.56 | 1.46 | 0.76 | 4.05 |
| RAML [6] | 11.26 | 3.10 | 11.65 | 1.92 | 5.35 | 1.76 | 1.09 | 5.16 |
| PCD | **7.23** | **2.20** | **5.66** | **0.99** | **2.86** | **0.89** | **0.74** | **2.93** |
| Δ | 0.74%↓ | 0.07%↓ | 1.88%↓ | 0.40%↓ | 0.15%↓ | 0.29%↓ | 0.02%↓ | 1.01%↓ |

| Method | CeFA (ACER ↓) | | | | | | | |
|---|---|---|---|---|---|---|---|---|
| | {R} | {D} | {I} | {R,D} | {R,I} | {D,I} | {R,D,I} | Average |
| Traditional [49] | 50.00 | 50.00 | 49.96 | 49.25 | 47.28 | 48.95 | 39.62 | 47.86 |
| Separate Model [49] | 27.44 | 33.75 | 36.17 | 35.62 | 31.62 | 36.62 | 24.15 | 32.20 |
| Augmentation [1] | 27.93 | 36.90 | 36.14 | 32.10 | 28.47 | 35.12 | 31.87 | 32.65 |
| HeMIS [15] | 34.14 | 37.97 | 36.94 | 36.02 | 33.94 | 31.92 | 40.66 | 35.94 |
| MMFormer [50] | 28.51 | 33.58 | 39.56 | 29.47 | 27.66 | 32.17 | 30.72 | 31.52 |
| MMANET [46] | 27.15 | 32.50 | 35.62 | 22.87 | 23.27 | 30.45 | 23.68 | 27.94 |
| MD [12] | 27.13 | 35.81 | 37.99 | 26.25 | 31.29 | 34.69 | 30.49 | 31.95 |
| ETMC [14] | 24.74 | 34.28 | 37.62 | 22.52 | 24.25 | 30.63 | 21.59 | 27.95 |
| RAML [6] | 28.54 | 33.88 | 40.01 | 23.82 | 28.81 | 28.85 | 22.11 | 29.43 |
| PCD | **21.38** | **28.01** | **34.79** | **17.19** | **20.92** | **21.68** | **14.39** | **22.63** |
| Δ | 3.36%↓ | 4.49%↓ | 0.83%↓ | 5.33%↓ | 2.35%↓ | 5.75%↓ | 7.20%↓ | 5.31%↓ |

| Method | NYUv2 (mIOU ↑) | | | | Cityscapes (mIOU ↑) | | | |
|---|---|---|---|---|---|---|---|---|
| | {R} | {D} | {R,D} | Average | {R} | {D} | {R,D} | Average |
| Traditional [36] | 11.15 | 4.18 | 48.78 | 21.41 | 3.17 | 4.87 | 78.73 | 28.89 |
| Separate Model [36] | 44.22 | 40.55 | 48.89 | 44.55 | 77.60 | 59.11 | 78.62 | 71.77 |
| Augmentation [1] | 41.34 | 39.76 | 47.23 | 42.77 | 76.89 | 57.42 | 78.13 | 70.81 |
| MMFormer [50] | 43.22 | 41.12 | 48.45 | 44.26 | 76.62 | 58.53 | 78.01 | 71.05 |
| MMANET [46] | 44.93 | 42.75 | **49.62** | 45.58 | 77.61 | 60.12 | 78.89 | 72.20 |
| PCD | **45.68** | **44.34** | 49.44 | **46.49** | **78.26** | **61.30** | **79.53** | **73.03** |
| Δ | 0.75%↑ | 1.59%↑ | 0.18%↓ | 0.91%↑ | 0.65%↑ | 1.18%↑ | 0.64%↑ | 0.83%↑ |

RGB-D dataset designed for urban scene comprehension. There are 5,000 annotated samples, where 2,975 samples are for training, 500 for validation, and 1,525 for testing.

**Experimental Details.** For classification CASIA-SURF and CeFA, the SGD optimizer [34] is used and the batch size is 64. The dimension of the Gaussian distribution is 512. We report the results using the metric of Average Classification Error Rate (ACER). Each modality leverages a separate ResNet-18 [17] as the unimodal encoder. We employ an exponential decay learning rate strategy in which the learning rate is fixed at 1e-3 during the warm-up stage and then decays exponentially. Weight decay and momentum are set to 0.0005 and 0.9, respectively. For segmentation experiments on NYUv2 and Cityscapes, we use the Adam optimizer [25] and set the batch size to 16. The results are evaluated by the metric of mean IOU (mIOU). The learning rate is initialized with 1e-2 and 1e-4 respectively for two datasets and adapted by the one-cycle scheduler. Following [46], we use ESANet [36] as the backbone. On all datasets, the variances are obtained through a two-layer MLP, where the hidden size is 1024. During training, we augment each modality-complete data by simulating all potential modality-missing scenarios and randomly sample one of the augmented data as the training sample for the current epoch. For bimodal datasets, three cases are included, that is, missing RGB, missing depth, and complete. For trimodal datasets, there are seven missing cases.

## 4.2 Performance Comparison

To evaluate the robustness of PCD, we choose the following methods in the comparison: 1) Baselines. Traditional [49, 36]: a benchmark method trained solely on modality-complete data. Separate Model [49, 36]: separate intermediate-fusion models for each modality combination. 2) Redundancy-based methods: Augmentation [1], MMFormer [50]. 3) Cross-modal KD-based methods: MMIN [51], MMANET [46]. 4) Dynamical fusion-based methods: MD [12], ETMC [14], RAML [6].

**Classification Task.** The results in Table 1 show the performance of PCD and other state-of-the-art (SOTA) methods across various testing conditions with missing modalities on two classification datasets CASIA-SURF and CeFA. We can see that the 'Traditional' method, which is exclusively trained on modality-complete samples, exhibits a high sensitivity to the missing modality problem. Specifically, the error rate surges by 21.63% on CASIA-SURF when only the RGB modality is available. Comparing the results of various missing modality methods, PCD achieves the best results in almost all the settings on the two multimodal classification datasets. In comparison to the second-best method, PCD demonstrates the error rate reductions of 1.01% and 5.31% on CASIA-SURF and CeFA. These results illustrate the effectiveness of our proposed method in privileged information transfer. Besides, the performance of some methods declines with an increasing number of modalities. For example, on CeFA, the error rate of MMANET with complete modalities is 0.81% higher than when IR is absent. This deterioration may potentially caused by overfitting resulting from deterministic alignment. In contrast, our method employs a probabilistic distillation, which introduces a more relaxed framework for aligning representations, mitigating this issue effectively.

**Segmentation Task.** We conduct experiments on NYUv2 and Cityscapes to verify the effectiveness of PCD on segmentation tasks. Compared to the second-best method, PCD achieves average accuracy improvements of 0.91% and 0.83% on NYUv2 and Cityscapes, respectively. Furthermore, in the Depth-missing scenarios on the NYUv2 and Cityscapes datasets, PCD demonstrates relatively small improvements. This may be because that the performance of the input RGB is already very close to that of the modality-complete input. Consequently, it is challenging to obtain additional privileged information through distillation, limiting the potential enhancement.

## 4.3 Further Analysis

**Ablation on Loss Components.** In this part, we investigate the impact of each loss component in Eq. (10) on CeFA. In Table 2, we conduct the ablation study and summarize the corresponding performance with or without different loss components. According to the results in Table 2, we can observe that the classi-

Table 2: Ablation study on CeFA. $\times$ and $\checkmark$ in the table indicate without and with the corresponding loss term respectively.

| $\mathcal{L}_c$ | $\mathcal{L}_u$ | $\mathcal{L}_g$ | {R} | {D} | {I} | {R,D} | {R,I} | {D,I} | {R,D,I} | Average |
|---|---|---|---|---|---|---|---|---|---|---|
| $\checkmark$ | $\times$ | $\times$ | 26.95 | 38.06 | 37.06 | 24.18 | 24.75 | 32.82 | 25.38 | 29.89 |
| $\checkmark$ | $\checkmark$ | $\times$ | 21.14 | 33.76 | 37.22 | 21.28 | 23.61 | 27.56 | 21.19 | 26.53 |
| $\checkmark$ | $\times$ | $\checkmark$ | **20.62** | 34.43 | 35.23 | 18.18 | 21.86 | 32.63 | 21.72 | 26.38 |
| $\checkmark$ | $\checkmark$ | $\checkmark$ | 21.38 | **28.01** | **34.79** | **17.19** | **20.92** | **21.68** | **14.39** | **22.63** |

fication model with the probability extremum loss $\mathcal{L}_u$ performs 3.36% better than the simple model with only $\mathcal{L}_c$, which suggests that constraining probabilities of extreme points indeed helps to the privileged information transfer from the modality-complete teacher to the modality-missing student. Additionally, PCD with all loss components outperforms the model with $\mathcal{L}_c$ and $\mathcal{L}_u$ on average, which validates the effectiveness of the geometric consistency loss.

**Ablation on Probabilistic Distillation.** To study the effect of probabilistic distillation, we conduct experiments to compare the performance of PCD with its determinate distillation variant. Here, the variant is the degradation method of PCD that transfers knowledge by directly minimizing the Euclidean distance of the complete-incomplete pairs in teacher and student networks. The results are shown in Table 3. It can be seen that PCD consistently

Table 3: The comparison between PCD and its variants on CeFA, where "Determinate" means the degradation of PCD with determinate distillation, while "Pretrained" is the distillation with a pretrained teacher.

| Configurations | {R} | {D} | {I} | {R,D} | {R,I} | {D,I} | {R,D,I} | Average |
|---|---|---|---|---|---|---|---|---|
| Determinate | 23.52 | 38.96 | 38.95 | 25.75 | 24.52 | 36.1 | 28.21 | 30.99 |
| Pretrained | 23.52 | 31.64 | 39.86 | 22.57 | 24.89 | 29.43 | 26.50 | 28.34 |
| PCD | **21.38** | **28.01** | **34.79** | **17.19** | **20.92** | **21.68** | **14.39** | **22.63** |

outperforms its "Determinate" variant in all missing modality combinations and decreases the error rate by 8.36% on average. This demonstrates the effectiveness of transferring privileged information via probabilistic distillation, which is more tolerant.

**Analysis about KD Strategy.** To explore the effectiveness of self-KD, we compare PCD with its pretrained teacher variant. This variant refers to training a modality-complete teacher individually to guide students in optimizing from scratch. The results are shown in Table 3. As can be seen, the error rate of PCD is 5.71% lower on average than its pretrained variant. In addition to training a fixed teacher to offer modality-complete supervision, our self-KD strategy also provides a good initialization for the student. With the help of the shared predictor, the semantic coherence of modality-complete and modality-missing representations is indirectly ensured, which narrows information gap between them at the beginning of KD, thereby facilitating privileged information transfer.

**Classification Boundary of the Teacher and Student.** In order to further validate the effectiveness of probabilistic distillation for the transfer of privileged complete-modality information, we analyze the predictions of both the fixed teacher obtained from the warm-up stage and the distilled student under all multimodal conditions. The results are shown in Figure 3. It can be observed that, apart from the reduced error rate, the logits of the student exhibit a higher concentration around 0 or 1, demonstrating a more separable inter-class boundary. The probabilistic distillation process transfers privileged information to hard samples around the classification boundary in a more tolerant way, mitigating the erroneous fit to spurious factors, so as to further refine modality-missing features.

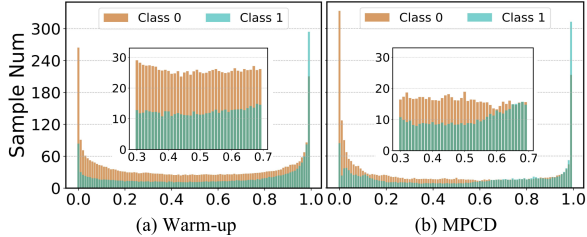

Figure 3: The prediction distributions of both the teacher and the distilled student of PCD under all multimodal combinations on CeFA. The X-axis represents the normalized logit output and the Y-axis is the number of samples after taking the square root.

**Hyperparameter $\lambda$.** The hyperparameter $\lambda$ controls the balance between distillation and classification. To validate the stability of PCD against $\lambda$, we conducted several experiments with different values of $\lambda$ on CeFA. The results are shown in the left half of Figure 4, where values of $\lambda$ range from 1.4 to 2.4. From the curve, we can see that setting a relatively large value for $\lambda$ enhances the distillation of privileged information, thereby enhancing the multimodal robustness. Specifically, in [1.4, 2.2], $\lambda$

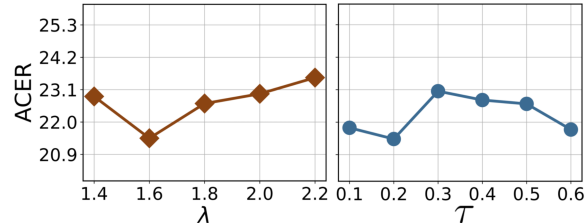

Figure 4: The average performance of PCD under different $\lambda$ and $\tau$ values on CeFA. The hyperparameter $\lambda$ is used to balance the loss terms, $\tau$ is the temperature.

appears to be insensitive within a certain range, In our experiments, we set $\lambda = 1.8$.

**Hyperparameter $\tau$.** In the right panel of Figure 4, we conducted several experiments with different values of $\tau$ to assess its impact on our results. The hyperparameters $\tau$ is the temperature in Equation (8), which scales the similarity measures. The results reveal $\tau$ is insensitive within a certain range. In our experiments, we set $\tau = 0.5$.

**Computational Overhead.** Compared to the multimodal models with the same backbone, PCD only introduces *a few additional head modules* in the encoder to estimate the variance. To demonstrate the minor change PCD brings, we estimated the number of parameters and FLOPs of PCD and the other three late fusion methods in Table 4. It can be seen that PCD does not significantly increase the number of parameters or FLOPs, where the FLOPs are almost equal to

Table 4: The numbers of parameters (M) and FLOPs (G) of several methods on CeFA.

| Method | Backbone | Paramters | FLOPs |
|---|---|---|---|
| MD [12] | ResNet-18 | 35.88 | 1.392 |
| ETMC [14] | ResNet-18 | 34.30 | 1.391 |
| RAML [6] | ResNet-18 | 35.09 | 1.393 |
| PCD | ResNet-18 | 38.50 | 1.395 |

the second-best method MMANET, while the number of parameters only increased by 4.20M. This lightweight change of MPCD makes it easily be applied to many existing multimodal fusion methods.

Table 5: Performance under different multimodal conditions when each unimodal data of training samples is missing with a probability of 30%.

| Method | {R} | {D} | {I} | {R,D} | {R,I} | {D,I} | {R,D,I} | Average |
|---|---|---|---|---|---|---|---|---|
| MMANET [46] | 28.39 | 39.61 | **34.12** | 34.19 | 23.39 | 34.12 | 27.11 | 31.56 |
| ETMC [14] | 25.96 | 34.69 | 38.60 | 24.15 | 24.58 | 31.83 | 24.03 | 29.12 |
| PCD | **23.42** | **30.23** | 34.60 | **18.34** | **21.98** | **24.50** | **15.07** | **24.02** |
| Δ | 2.54%↓ | 4.46%↑ | 0.48%↑ | 5.81%↓ | 1.41%↓ | 6.43%↓ | 8.96%↓ | 5.20%↓ |

**Modality-Missing Training Data.** All the experiments above are conducted with the modality-complete training data. In this part, we extend PCD by considering the scenario where the modality-complete data of some training samples is also unavailable. PCD is only applied to the data that has modality-complete counterpart, and for the remaining data, only $\mathcal{L}_t$ is optimized. Here, we introduce a case where 30% of the data is consistently missing from each modality during training. As shown in Table 5, while some modality-missing cases may underperform compared to the SOTA, PCD still outperforms the second-best method by 5.20% on average. Although PCD is not specifically designed for modality-missing training data, these results demonstrate its scalability for such scenarios.

# 5 Conclusion

In this paper, we propose a multimodal Probabilistic Distillation (PCD) method to mitigate the missing modality problem, which considers the indeterminacy in the alignment between the modality-complete and modality-missing representations. Specifically, PCD aims to parameterize the modality-missing representations as different Gaussian distributions and fit PDFs of their mapped variables in the modality-complete space. This is achieved by ensuring the characteristics of probabilities at extreme points and maintaining geometric consistency with that of the modality-complete features. Extensive experiments validate the superiority of PCD in increasing multimodal robustness.

# Acknowledgement

This work is supported by the National Key R&D Program of China (No. 2022ZD0160702), STCSM (No. 22511106101, No. 22DZ2229005), 111 plan (No. BP0719010) and National Natural Science Foundation of China (No. 62306178).

# References

[1] Michal Bednarek, Piotr Kicki, and Krzysztof Walas. On robustness of multi-modal fusion—robotics perspective. *Electronics*, 9(7):1152, 2020.

[2] Lei Cai, Zhengyang Wang, Hongyang Gao, Dinggang Shen, and Shuiwang Ji. Deep adversarial learning for multi-modality missing data completion. In *Proceedings of the 24th ACM SIGKDD international conference on knowledge discovery & data mining*, pages 1158–1166, 2018.

[3] Jinming Cao, Hanchao Leng, Dani Lischinski, Daniel Cohen-Or, Changhe Tu, and Yangyan Li. Shapeconv: Shape-aware convolutional layer for indoor rgb-d semantic segmentation. In *Proceedings of the IEEE/CVF international conference on computer vision*, pages 7088–7097, 2021.

[4] Jie Chang, Zhonghao Lan, Changmao Cheng, and Yichen Wei. Data uncertainty learning in face recognition. In *Proceedings of the IEEE/CVF conference on computer vision and pattern recognition*, pages 5710–5719, 2020.

[5] Mengxi Chen, Linyu Xing, Yu Wang, and Ya Zhang. Enhanced multimodal representation learning with cross-modal kd. In *Proceedings of the IEEE/CVF Conference on Computer Vision and Pattern Recognition*, pages 11766–11775, 2023.

[6] Mengxi Chen, Jiangchao Yao, Linyu Xing, Yu Wang, Ya Zhang, and Yanfeng Wang. Redundancy-adaptive multimodal learning for imperfect data. *arXiv preprint arXiv:2310.14496*, 2023.

[7] Yanbei Chen, Yongqin Xian, A Koepke, Ying Shan, and Zeynep Akata. Distilling audio-visual knowledge by compositional contrastive learning. In *Proceedings of the IEEE/CVF Conference on Computer Vision and Pattern Recognition*, pages 7016–7025, 2021.

[8] Marius Cordts, Mohamed Omran, Sebastian Ramos, Timo Rehfeld, Markus Enzweiler, Rodrigo Benenson, Uwe Franke, Stefan Roth, and Bernt Schiele. The cityscapes dataset for semantic urban scene understanding. In *Proceedings of the IEEE conference on computer vision and pattern recognition*, pages 3213–3223, 2016.

[9] Yuhang Ding, Xin Yu, and Yi Yang. Rfnet: Region-aware fusion network for incomplete multi-modal brain tumor segmentation. In *Proceedings of the IEEE/CVF international conference on computer vision*, pages 3975–3984, 2021.

[10] Tiantian Feng, Daniel Yang, Digbalay Bose, and Shrikanth Narayanan. Can text-to-image model assist multi-modal learning for visual recognition with visual modality missing? *arXiv preprint arXiv:2402.09036*, 2024.

[11] Nuno C Garcia, Pietro Morerio, and Vittorio Murino. Modality distillation with multiple stream networks for action recognition. In *Proceedings of the European Conference on Computer Vision (ECCV)*, pages 103–118, 2018.

[12] Zongbo Han, Fan Yang, Junzhou Huang, Changqing Zhang, and Jianhua Yao. Multimodal dynamics: Dynamical fusion for trustworthy multimodal classification. In *Proceedings of the IEEE/CVF Conference on Computer Vision and Pattern Recognition*, pages 20707–20717, 2022.

[13] Zongbo Han, Changqing Zhang, Huazhu Fu, and Joey Tianyi Zhou. Trusted multi-view classification. *arXiv preprint arXiv:2102.02051*, 2021.

[14] Zongbo Han, Changqing Zhang, Huazhu Fu, and Joey Tianyi Zhou. Trusted multi-view classification with dynamic evidential fusion. *IEEE transactions on pattern analysis and machine intelligence*, 45(2):2551–2566, 2022.

[15] Mohammad Havaei, Nicolas Guizard, Nicolas Chapados, and Yoshua Bengio. Hemis: Hetero-modal image segmentation. In *Medical Image Computing and Computer-Assisted Intervention–MICCAI 2016: 19th International Conference, Athens, Greece, October 17-21, 2016, Proceedings, Part II 19*, pages 469–477. Springer, 2016.

[16] Kaiming He, Haoqi Fan, Yuxin Wu, Saining Xie, and Ross Girshick. Momentum contrast for unsupervised visual representation learning. In *Proceedings of the IEEE/CVF Conference on Computer Vision and Pattern Recognition*, June 2020.

[17] Kaiming He, Xiangyu Zhang, Shaoqing Ren, and Jian Sun. Deep residual learning for image recognition. In *Proceedings of the IEEE conference on computer vision and pattern recognition*, pages 770–778, 2016.

[18] Yu Huang, Chenzhuang Du, Zihui Xue, Xuanyao Chen, Hang Zhao, and Longbo Huang. What makes multi-modal learning better than single (provably). *Advances in Neural Information Processing Systems*, 34:10944–10956, 2021.

[19] Ziqi Huang, Li Lin, Pujin Cheng, Linkai Peng, and Xiaoying Tang. Multi-modal brain tumor segmentation via missing modality synthesis and modality-level attention fusion. *arXiv preprint arXiv:2203.04586*, 2022.

[20] Wen-Da Jin, Jun Xu, Qi Han, Yi Zhang, and Ming-Ming Cheng. Cdnet: Complementary depth network for rgb-d salient object detection. *IEEE Transactions on Image Processing*, 30:3376–3390, 2021.

[21] Vijay John and Yasutomo Kawanishi. A multimodal sensor fusion framework robust to missing modalities for person recognition. In *Proceedings of the 4th ACM International Conference on Multimedia in Asia*, pages 1–5, 2022.

[22] Jiang Jue, Hu Jason, Tyagi Neelam, Rimner Andreas, Berry L Sean, Deasy O Joseph, and Veeraraghavan Harini. Integrating cross-modality hallucinated mri with ct to aid mediastinal lung tumor segmentation. In *Medical Image Computing and Computer Assisted Intervention–MICCAI 2019: 22nd International Conference, Shenzhen, China, October 13–17, 2019, Proceedings, Part VI 22*, pages 221–229. Springer, 2019.

[23] Prannay Khosla, Piotr Teterwak, Chen Wang, Aaron Sarna, Yonglong Tian, Phillip Isola, Aaron Maschinot, Ce Liu, and Dilip Krishnan. Supervised contrastive learning. *Advances in neural information processing systems*, 33:18661–18673, 2020.

[24] Kyungyul Kim, ByeongMoon Ji, Doyoung Yoon, and Sangheum Hwang. Self-knowledge distillation with progressive refinement of targets. In *Proceedings of the IEEE/CVF international conference on computer vision*, pages 6567–6576, 2021.

[25] Diederik P Kingma and Jimmy Ba. Adam: A method for stochastic optimization. *arXiv preprint arXiv:1412.6980*, 2014.

[26] Yi-Lun Lee, Yi-Hsuan Tsai, Wei-Chen Chiu, and Chen-Yu Lee. Multimodal prompting with missing modalities for visual recognition. In *Proceedings of the IEEE/CVF Conference on Computer Vision and Pattern Recognition*, pages 14943–14952, 2023.

[27] Ajian Liu, Zichang Tan, Jun Wan, Sergio Escalera, Guodong Guo, and Stan Z Li. Casia-surf cefa: A benchmark for multi-modal cross-ethnicity face anti-spoofing. In *Proceedings of the IEEE/CVF Winter Conference on Applications of Computer Vision*, pages 1179–1187, 2021.

[28] Ajian Liu, Zichang Tan, Jun Wan, Yanyan Liang, Zhen Lei, Guodong Guo, and Stan Z Li. Face anti-spoofing via adversarial cross-modality translation. *IEEE Transactions on Information Forensics and Security*, 16:2759–2772, 2021.

[29] Ajian Liu, Jun Wan, Sergio Escalera, Hugo Jair Escalante, Zichang Tan, Qi Yuan, Kai Wang, Chi Lin, Guodong Guo, Isabelle Guyon, et al. Multi-modal face anti-spoofing attack detection challenge at cvpr2019. In *Proceedings of the IEEE/CVF Conference on Computer Vision and Pattern Recognition Workshops*, pages 0–0, 2019.

[30] Haojie Liu, Shun Ma, Daoxun Xia, and Shaozi Li. Sfanet: A spectrum-aware feature augmentation network for visible-infrared person reidentification. *IEEE Transactions on Neural Networks and Learning Systems*, 2021.

[31] Mengmeng Ma, Jian Ren, Long Zhao, Davide Testuggine, and Xi Peng. Are multimodal transformers robust to missing modality? In *Proceedings of the IEEE/CVF Conference on Computer Vision and Pattern Recognition*, pages 18177–18186, 2022.

[32] Mengmeng Ma, Jian Ren, Long Zhao, Sergey Tulyakov, Cathy Wu, and Xi Peng. Smil: Multimodal learning with severely missing modality. In *Proceedings of the AAAI Conference on Artificial Intelligence*, volume 35, pages 2302–2310, 2021.

[33] Harsh Maheshwari, Yen-Cheng Liu, and Zsolt Kira. Missing modality robustness in semi-supervised multi-modal semantic segmentation. In *Proceedings of the IEEE/CVF Winter Conference on Applications of Computer Vision*, pages 1020–1030, 2024.

[34] Herbert Robbins and Sutton Monro. A stochastic approximation method. *The annals of mathematical statistics*, pages 400–407, 1951.

[35] Enrique Sanchez, Mani Kumar Tellamekala, Michel Valstar, and Georgios Tzimiropoulos. Affective processes: stochastic modelling of temporal context for emotion and facial expression recognition. In *Proceedings of the IEEE/CVF Conference on Computer Vision and Pattern Recognition*, pages 9074–9084, 2021.

[36] Daniel Seichter, Mona Köhler, Benjamin Lewandowski, Tim Wengefeld, and Horst-Michael Gross. Efficient rgb-d semantic segmentation for indoor scene analysis. In *2021 IEEE international conference on robotics and automation (ICRA)*, pages 13525–13531. IEEE, 2021.

[37] Leslie N Smith and Nicholay Topin. Super-convergence: Very fast training of neural networks using large learning rates. In *Artificial intelligence and machine learning for multi-domain operations applications*, volume 11006, pages 369–386. SPIE, 2019.

[38] Shuran Song, Samuel P Lichtenberg, and Jianxiong Xiao. Sun rgb-d: A rgb-d scene understanding benchmark suite. In *Proceedings of the IEEE conference on computer vision and pattern recognition*, pages 567–576, 2015.

[39] Jonathan Stroud, David Ross, Chen Sun, Jia Deng, and Rahul Sukthankar. D3d: Distilled 3d networks for video action recognition. In *Proceedings of the IEEE/CVF Winter Conference on Applications of Computer Vision*, pages 625–634, 2020.

[40] Luan Tran, Xiaoming Liu, Jiayu Zhou, and Rong Jin. Missing modalities imputation via cascaded residual autoencoder. In *Proceedings of the IEEE conference on computer vision and pattern recognition*, pages 1405–1414, 2017.

[41] Hu Wang, Yuanhong Chen, Congbo Ma, Jodie Avery, Louise Hull, and Gustavo Carneiro. Multi-modal learning with missing modality via shared-specific feature modelling. In *Proceedings of the IEEE/CVF Conference on Computer Vision and Pattern Recognition*, pages 15878–15887, 2023.

[42] Hu Wang, Congbo Ma, Yuyuan Liu, Yuanhong Chen, Yu Tian, Jodie Avery, Louise Hull, and Gustavo Carneiro. Enhancing multi-modal learning: Meta-learned cross-modal knowledge distillation for handling missing modalities. *arXiv preprint arXiv:2405.07155*, 2024.

[43] Hu Wang, Congbo Ma, Jianpeng Zhang, Yuan Zhang, Jodie Avery, Louise Hull, and Gustavo Carneiro. Learnable cross-modal knowledge distillation for multi-modal learning with missing modality. In *International Conference on Medical Image Computing and Computer-Assisted Intervention*, pages 216–226. Springer, 2023.

[44] Qi Wang, Liang Zhan, Paul Thompson, and Jiayu Zhou. Multimodal learning with incomplete modalities by knowledge distillation. In *Proceedings of the 26th ACM SIGKDD International Conference on Knowledge Discovery & Data Mining*, pages 1828–1838, 2020.

[45] Shuai Wang, Zipei Yan, Daoan Zhang, Haining Wei, Zhongsen Li, and Rui Li. Prototype knowledge distillation for medical segmentation with missing modality. In *ICASSP 2023-2023 IEEE International Conference on Acoustics, Speech and Signal Processing (ICASSP)*, pages 1–5. IEEE, 2023.

[46] Shicai Wei, Chunbo Luo, and Yang Luo. Mmanet: Margin-aware distillation and modality-aware regularization for incomplete multimodal learning. In *Proceedings of the IEEE/CVF Conference on Computer Vision and Pattern Recognition*, pages 20039–20049, 2023.

[47] Jiandian Zeng, Tianyi Liu, and Jiantao Zhou. Tag-assisted multimodal sentiment analysis under uncertain missing modalities. In *Proceedings of the 45th International ACM SIGIR Conference on Research and Development in Information Retrieval*, pages 1545–1554, 2022.

[48] Fei Zhang, Tianfei Zhou, Boyang Li, Hao He, Chaofan Ma, Tianjiao Zhang, Jiangchao Yao, Ya Zhang, and Yanfeng Wang. Uncovering prototypical knowledge for weakly open-vocabulary semantic segmentation. *Advances in Neural Information Processing Systems*, 36:73652–73665, 2023.

[49] Shifeng Zhang, Xiaobo Wang, Ajian Liu, Chenxu Zhao, Jun Wan, Sergio Escalera, Hailin Shi, Zezheng Wang, and Stan Z Li. A dataset and benchmark for large-scale multi-modal face anti-spoofing. In *Proceedings of the IEEE/CVF Conference on Computer Vision and Pattern Recognition*, pages 919–928, 2019.

[50] Yao Zhang, Nanjun He, Jiawei Yang, Yuexiang Li, Dong Wei, Yawen Huang, Yang Zhang, Zhiqiang He, and Yefeng Zheng. mmformer: Multimodal medical transformer for incomplete multimodal learning of brain tumor segmentation. In *International Conference on Medical Image Computing and Computer-Assisted Intervention*, pages 107–117. Springer, 2022.

[51] Jinming Zhao, Ruichen Li, and Qin Jin. Missing modality imagination network for emotion recognition with uncertain missing modalities. In *Proceedings of the 59th Annual Meeting of the Association for Computational Linguistics and the 11th International Joint Conference on Natural Language Processing*, pages 2608–2618, 2021.

[52] Zihua Zhao, Mengxi Chen, Tianjie Dai, Jiangchao Yao, Bo Han, Ya Zhang, and Yanfeng Wang. Mitigating noisy correspondence by geometrical structure consistency learning. In *Proceedings of the IEEE/CVF Conference on Computer Vision and Pattern Recognition*, pages 27381–27390, 2024.

[53] Tongxue Zhou, Stéphane Canu, Pierre Vera, and Su Ruan. Brain tumor segmentation with missing modalities via latent multi-source correlation representation. In *Medical Image Computing and Computer Assisted Intervention–MICCAI 2020: 23rd International Conference, Lima, Peru, October 4–8, 2020, Proceedings, Part IV 23*, pages 533–541. Springer, 2020.

[54] Zhihan Zhou, Jiangchao Yao, Feng Hong, Ya Zhang, Bo Han, and Yanfeng Wang. Combating representation learning disparity with geometric harmonization. In *Thirty-seventh Conference on Neural Information Processing Systems*, 2023.

[55] Zhihan Zhou, Jiangchao Yao, Yan-Feng Wang, Bo Han, and Ya Zhang. Contrastive learning with boosted memorization. In *International Conference on Machine Learning*, pages 27367–27377. PMLR, 2022.

[56] Yizhe Zhu, Xin Sun, and Xi Zhou. Exploiting multi-modal fusion for robust face representation learning with missing modality. In *International Conference on Artificial Neural Networks*, pages 283–294. Springer, 2023.

# Appendix / Supplemental Material

## A  Algorithm

The whole training procedure of PCD is shown in Algorithm 1.

---
**Algorithm 1** Training procedure of PCD

---
**Input:** Training data $X$, target set $Y$, missing modality indicator $\delta$, hyperparameters $\lambda, \tau$, warm-up epoch $E$, training epoch $P$ and batch size $B$
// The warm-up stage
**for** an epoch $e = 1, \cdots, E$ **do**
    **for** a sampled batch $X = \{x_i\}_{i=1}^{B}$, $Y = \{y_i\}_{i=1}^{B}$ **do**
        Optimize $\mathcal{L}_c$        // Initialize the teacher and student models
    **end for**
**end for**
// The training stage
Load and fix parameters in warm-up stage for the teacher
**for** a epoch $p = 1, \cdots, P$ **do**
    **for** a sampled batch $X = \{x_i\}_{i=1}^{B}$, $Y = \{y_i\}_{i=1}^{B}$ **do**
        Calculate $z^\star$ and $g^\star$
        Model Gaussian distributions in Equation (6)
        Calculate the probability extremum loss $\mathcal{L}_u$ and the geometric consistency loss $\mathcal{L}_g$
        Optimize Equation (10) to update parameters of the student
    **end for**
**end for**

---

## B  Implement Details

### B.1  Classification

**Network Architecture.** For a fair comparison, we follow the basic implementation of the traditional multimodal model in [49] for all the comparison methods. This backbone is a late fusion network with separate ResNet18 encoders for each modality. Here, for PCD, we parameterize the unimodal features from unimodal encoders and the fused multimodal features from the fusion module as independent Gaussian distributions, and make them fit their PDFs by optimizing corresponding $\mathcal{L}_u$ and $\mathcal{L}_g$. The variance is obtained for a two-layer MLP, where the hidden size is 1024. In addition, like [6], by analyzing the variance in unimodal distributions, a weighting mechanism is employed, which can adaptively aggregate the information of each available unimodality.

**Setup.** We augment modality-complete samples by simulating all potential missing modality scenarios equally. In other words, in one epoch, each sample has an equal probability of randomly encountering one of seven missing modality scenarios. Besides, random flipping, rotation, and cropping are also used for data augmentation. All models are optimized by an SGD for 110 epochs with a mini-batch of 64. Weight decay and momentum are set to 0.0005 and 0.9, respectively. The learning rate is initialized to 0.001. After the warm-up stage, an exponential decay learning rate strategy is employed, in which the decay coefficient is 0.9. The dimension of the Gaussian distribution is 512. The hyper-parameters $\lambda, \tau$ are 1.8 and 0.5, respectively.

### B.2  Segmentation

**Network Architecture.** We use the ESANet [36] as the backbone, which is an early fusion network. The modality encoder is the ResNet50 with NBt1 used in ESANet. For PCD, we parameterize the fused multimodal features from the last three resolution stages as independent Gaussian distributions. Notice that, since the dimensionality of multimodal features is very high, only one negative vector in Equation (8) is selected to conserve computational resources, and this formulation degenerates to the triplet loss. Besides, $\mathcal{L}_u$ is applied to the fused features after average pooling.

**Setup.** Random flipping, rotation, cropping and missing modality simulation are used for data augmentation. All models are optimized by an Adam for 450 epochs with a mini-batch of 16. The

Table 6: Stability experiments on NYUv2, Cityscapes, CASIA-SURF and CeFA.

| | {R} | {D} | {I} | {R,D} | {R,I} | {D,I} | {R,D,I} | Average |
|---|---|---|---|---|---|---|---|---|
| | | | | CASIA-SURF | | | | |
| PCD | 7.23 ±0.13 | 2.20 ±0.26 | 5.66 ±0.90 | 0.99 ±0.10 | 2.86 ±0.31 | 0.89 ±0.19 | 0.74 ±0.23 | 2.93 ±0.25 |

| | {R} | {D} | {I} | {R,D} | {R,I} | {D,I} | {R,D,I} | Average |
|---|---|---|---|---|---|---|---|---|
| | | | | CeFA | | | | |
| PCD | 21.38 ±1.85 | 28.01 ±2.06 | 34.79 ±2.46 | 17.19 ±0.65 | 20.92 ±2.41 | 21.68 ±3.61 | 14.39 ±3.54 | 22.63 ±2.18 |

| | NYUv2 | | | | Cityscapes | | | |
|---|---|---|---|---|---|---|---|---|
| | {R} | {T} | {R,T} | Average | {R} | {T} | {R,T} | Average |
| PCD | 45.68 ±0.11 | 44.34 ± 0.08 | 49.44 ± 0.08 | 46.49 ±0.04 | 78.26 ±0.21 | 61.30 ±0.26 | 79.53 ±0.29 | 73.03 ±0.11 |

Table 7: Ablation study of loss components on CASIA-SURF, CeFA, NYUv2 and Cityscapes.

| $\mathcal{L}_c$ | $\mathcal{L}_u$ | $\mathcal{L}_g$ | {R} | {D} | {I} | {R,D} | {R,I} | {D,I} | {R,D,I} | Average |
|---|---|---|---|---|---|---|---|---|---|
| | | | | | | CASIA-SURF | | | |
| ✓ | × | × | 12.31 | 2.89 | 19.24 | 1.31 | 8.16 | 2.19 | 1.35 | 6.78 |
| ✓ | × | ✓ | 13.55 | **2.01** | 18.02 | **0.86** | 5.81 | 2.53 | 0.85 | 6.24 |
| ✓ | ✓ | × | 7.59 | 4.10 | 7.97 | 1.83 | 3.86 | 2.04 | 0.97 | 4.05 |
| ✓ | ✓ | ✓ | **7.23** | 2.20 | **5.66** | 0.99 | **2.86** | **0.89** | **0.74** | **2.93** |

| $\mathcal{L}_c$ | $\mathcal{L}_u$ | $\mathcal{L}_g$ | {R} | {D} | {I} | {R,D} | {R,I} | {D,I} | {R,D,I} | Average |
|---|---|---|---|---|---|---|---|---|---|
| | | | | | | CeFA | | | |
| ✓ | × | × | 26.95 | 38.06 | 37.06 | 24.18 | 24.75 | 32.82 | 25.38 | 29.89 |
| ✓ | ✓ | × | 21.14 | 33.76 | 37.22 | 21.28 | 23.61 | 27.56 | 21.19 | 26.53 |
| ✓ | × | ✓ | **20.62** | 34.43 | 35.23 | 18.18 | 21.86 | 32.63 | 21.72 | 26.38 |
| ✓ | ✓ | ✓ | 21.38 | **28.01** | **34.79** | **17.19** | **20.92** | **21.68** | **14.39** | **22.63** |

| $\mathcal{L}_c$ | $\mathcal{L}_u$ | $\mathcal{L}_g$ | {R} | {T} | {R,T} | Average | {R} | {T} | {R,T} | Average |
|---|---|---|---|---|---|---|---|---|---|
| | | | NYUv2 | | | | Cityscapes | | | |
| ✓ | × | × | 44.24 | 41.17 | 47.89 | 44.43 | 77.54 | 59.64 | 78.46 | 71.89 |
| ✓ | × | ✓ | 45.96 | 42.95 | 48.54 | 45.82 | 78.11 | 60.62 | 79.07 | 72.60 |
| ✓ | ✓ | × | 44.48 | 42.02 | 48.86 | 45.12 | 77.52 | 59.94 | 78.91 | 72.17 |
| ✓ | ✓ | ✓ | **45.68** | **44.34** | **49.44** | **46.49** | **78.26** | **61.30** | **79.53** | **73.03** |

learning rate is initialized to 0.01 and the warm-up epoch is set as 150. After the warm-up stage, a cosine annealing learning rate strategy is employed.

## C   Stability Experiments

In Table 6, we detail the stability experiments for PCD across all datasets. Each experiment is repeated for three times to ensure reliability, allowing to calculate the average score along with the standard deviation. The results reveal that, even in its worst-case scenario, PCD outperforms the best competing methods, registering average improvements of 0.87% on NYUv2, 0.72% on Cityscapes, 0.76% on CASIA-SURF, and a significant 3.13% on CeFA. These outcomes not only underscore PCD's superior performance but also attest to its stability and consistency across a wide range of testing conditions. This consistent reliability highlights the robustness and adaptability of PCD, making it an effective solution in varied scenarios.

## D   The Visualization of Feature distribution

We use t-SNE to visualize the distribution of the modality-complete, RGB, Depth, and IR representations of the unified model without PCD distillation on CASIA-SURF. The results are shown in

Figure 5. It can be observed that each unimodal distribution is similar but different to the modality-complete distribution, which provides empirical evidence for PCD to consider the indeterminacy in the mapping from incompleteness to completeness.

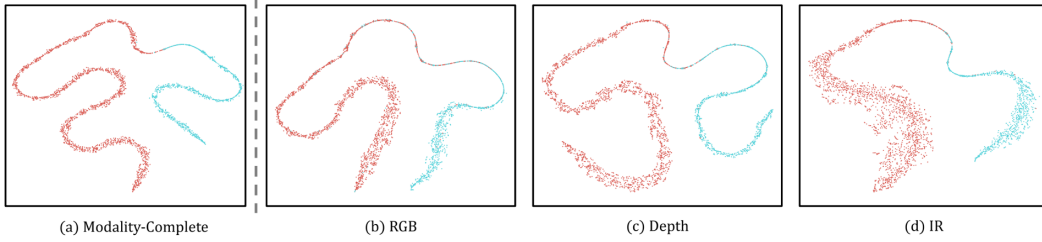

(a) Modality-Complete　　(b) RGB　　(c) Depth　　(d) IR

Figure 5: The visualization of the distributions of the modality-complete, RGB, Depth, and IR representations from the unified model without distillation.

# E  Ablation Study

## E.1  Ablation Study on Loss Components

We further conduct ablation studies to evaluate the effects of different loss components on the NYUv2, Cityscapes, CASIA-SURF, and CeFA datasets, as presented in Table 7. Notably, incorporating any of the loss components yields substantial improvements, particularly with the CeFA dataset. When applied separately, $\mathcal{L}_u$ and $\mathcal{L}_g$ each contributed to an average accuracy improvement of 3.36% and 3.51% respectively. These results underscore the significance of constraining probabilities of extreme points for enhancing the transfer of privileged information. Overall, the PCD model achieves optimal performance when it incorporates all proposed loss components.

## E.2  Analysis of Hyperparameter $\lambda$

To further assess the stability of the PCD model in response to various $\lambda$ parameters, we report its average performance across the CASIA-SURF and CeFA datasets, as illustrated in the left panel of Figure 6. The performance curve demonstrates that PCD maintains considerable stability across a range of $\lambda$ values, where the performance variance is kept within 0.8. Notably, PCD consistently outperforms SOTA models on all datasets when the $\lambda$ value is between 1.6 and 2. Based on these observations, we have set $\lambda$ to 1.8 throughout our classification experiments to ensure optimal performance and stability. This consistent outperformance underscores the robustness of the PCD model under varying conditions.

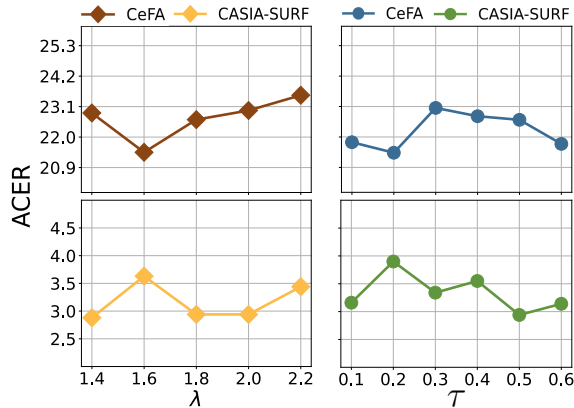

Figure 6: The average performance of PCD under different $\lambda$ and $\tau$ values on CASIA-SURF and CeFA.

## E.3  Analysis of Hyperparameter $\tau$

In the right panel of Figure 6, we conducted a series of experiments to evaluate the impact of different values of the hyperparameter $\tau$ on the performance of PCD on the multimodal classification datasets CASIA-SURF and CeFA. This hyperparameter, which acts as the temperature coefficient in Equation (8), is used to scale the similarity measures. The experimental findings indicate that the performance of the model is relatively insensitive to variations in $\tau$ within a certain range. Based on our results, we chose to set $\tau$ to 0.5 for all subsequent experiments to ensure an optimal balance between performance and parameter sensitivity.

Table 8: Analysis of warm-up epoch on CeFA.

| Epoch | {R} | {D} | {I} | {R,D} | {R,I} | {D,I} | {R,D,I} | Avg |
|-------|------|------|------|-------|-------|-------|---------|------|
| 30 | **19.28** | 32.35 | 36.41 | 17.15 | **20.26** | 26.26 | 16.18 | 23.98 |
| 40 | 19.29 | 28.20 | 36.40 | **15.30** | 20.58 | 24.08 | 17.71 | 23.08 |
| 50 | 21.38 | 28.01 | **34.79** | 17.19 | 20.92 | 21.68 | **14.39** | **22.63** |
| 60 | 21.46 | **27.16** | 35.17 | 16.75 | 22.97 | 22.35 | 15.21 | 23.01 |
| 70 | 21.64 | 29.22 | 35.86 | 17.90 | 21.69 | 23.51 | 17.16 | 23.86 |
| 80 | 23.35 | 26.38 | 33.83 | 16.40 | 25.25 | **21.14** | 19.83 | 23.74 |

### E.4 Analysis of Warm-up

Warm-up stage learns to provide complete modality supervision and a good initialization for the subsequent training process. In this part, we investigate the impact of varying warm-up epochs on probabilistic distillation. The experimental results in Table 8 emphasize the importance of judiciously setting the warm-up epoch. The experimental results show that PCD is not sensitive to the number of warm-up epochs. Within the range of 30 to 80, the average result is around 23%, consistently outperforming the SOTA. We set the number of warm-up epochs to 50 for the classification tasks.

## F Results on SUN RGB-D Dataset

To further confirm the effectiveness of PCD on segmentation tasks, we conduct experiments on a larger dataset, SUN RGB-D [38]. This dataset has 37 categories of objects and contains 5,285 RGB-Depth pairs for training and 5050 pairs for testing. The results are shown in Table 9. We can see that PCD is effective even on a larger segmented dataset.

Table 9: The mIOU($\uparrow$) of PCD and other methods on SUN RGB-D.

| Methods | {R} | {D} | {R,T} | Average |
|---------|------|------|-------|---------|
| Separate Model | 43.94 | 39.81 | 47.84 | 43.86 |
| MMANET | 44.73 | 39.94 | **47.54** | 44.07 |
| PCD | $45.63_{\pm0.16}$ | $41.43_{\pm0.07}$ | $47.24_{\pm0.17}$ | $44.75_{\pm0.02}$ |

## G Exploration on Modality-Missing Training Data

In Table 10, we conduct experiments on PCD against multiple SOTAs on the scenarios of training data with missing modalities. Specifically, we evaluated the performance on both the CASIA-SURF and CeFA datasets, where each modality of the training data has either 30% or 40% of its data missing. The results clearly indicate that PCD outperforms all other methods at both rates. Notably, PCD shows a significant performance improvement on CeFA, with a gap of 5.39% under the 30% missing modality condition and 5.10% under the 40% missing modality condition. These results indicate that although PCD is not specifically designed for modality-missing training data, it is still scalable for this scenario.

## H Limitations and Future Explorations

This paper introduces a probabilistic alignment approach between modality-complete and modality-missing representations to enhance the effective transfer of privileged information. The proposed method is primarily designed for scenarios where all training samples are modality-complete, and modality-missing occurs exclusively during testing. If modality-missing data is present during training, knowledge distillation cannot be applied to the modality-missing subset of the data. Therefore, in the future, scenarios with missing data during training will be further the focus of our consideration.

Table 10: Performance under different training data missing modality rates. The best results are in bold and the second-best ones are marked with underline. "Δ" means the performance gap between PCD and the second-best results.

| Missing | Method | {R} | {D} | {I} | {R,D} | {R,I} | {D,I} | {R,D,I} | Average |
|---|---|---|---|---|---|---|---|---|---|
| | | CASIA-SURF (ACER ↓) | | | | | | | |
| 30% | MMANET [46] | 13.50 | 3.38 | 6.57 | 6.57 | 3.72 | 1.83 | 1.31 | 4.67 |
| | ETMC [14] | 7.63 | 3.62 | 10.18 | 1.12 | 5.21 | 1.43 | 0.96 | 4.31 |
| | PCD | 8.28 | 2.13 | 6.66 | 1.24 | 2.66 | 2.66 | 0.60 | 3.18 |
| | Δ | 0.65%↑ | 1.25%↓ | 0.09%↑ | 0.12%↑ | 1.06%↓ | 1.23%↑ | 0.36%↓ | 1.13%↓ |
| 40% | MMANET [46] | 14.96 | 5.22 | 9.03 | 3.24 | 5.14 | 2.31 | 2.10 | 6.00 |
| | ETMC [14] | 9.38 | 7.42 | 7.44 | 1.41 | 3.98 | 3.16 | 0.58 | 4.77 |
| | PCD | 7.14 | 1.77 | 10.88 | 1.08 | 3.70 | 1.10 | 0.88 | 3.79 |
| | Δ | 2.24%↓ | 3.45%↓ | 3.44%↑ | 0.33%↓ | 0.28%↓ | 1.21%↓ | 0.30%↑ | 0.98%↓ |
| | | CeFA (ACER ↓) | | | | | | | |
| 30% | MMANET [46] | 28.39 | 39.61 | 34.12 | 34.19 | 23.39 | 34.12 | 27.11 | 31.56 |
| | ETMC [14] | 25.96 | 34.69 | 38.60 | 24.15 | 24.58 | 31.83 | 24.03 | 29.12 |
| | PCD | 23.42 | 30.23 | 34.60 | 18.34 | 21.98 | 24.50 | 15.07 | 23.73 |
| | Δ | 2.54%↓ | 4.46%↓ | 0.48%↑ | 5.81%↓ | 1.41%↓ | 7.33%↓ | 8.96%↓ | 5.39%↓ |
| 40% | MMANET [46] | 29.94 | 43.40 | 37.29 | 31.60 | 28.62 | 44.97 | 31.80 | 35.38 |
| | ETMC [14] | 24.38 | 37.82 | 38.33 | 25.04 | 24.39 | 36.96 | 24.03 | 30.13 |
| | PCD | 24.91 | 31.23 | 34.40 | 21.09 | 23.98 | 23.31 | 16.30 | 25.03 |
| | Δ | 0.53%↑ | 6.58%↓ | 2.89%↓ | 3.95%↓ | 0.40%↓ | 13.65%↓ | 7.73%↓ | 5.10%↓ |

# I Impact Statements

The method proposed in this paper can effectively improve the robustness of the multimodal model. This exploration is of great significance to the real-world inference scenarios that can not always obtain modality-complete data, such as healthcare and automatic driving. Compared to previous methods, PCD does not add a lot of parameters and effectively saves computational costs. So far, we have not discovered any negative impacts of this method.

